# Improved Hidden Markov Model Speech Recognition Using Radial Basis Function Networks

Elliot Singer and Richard P. Lippmann
Lincoln Laboratory, MIT
Lexington, MA 02173-9108, USA

## Abstract

A high performance speaker-independent isolated-word hybrid speech recognizer was developed which combines Hidden Markov Models (HMMs) and Radial Basis Function (RBF) neural networks. In recognition experiments using a speaker-independent E-set database, the hybrid recognizer had an error rate of 11.5% compared to 15.7% for the robust unimodal Gaussian HMM recognizer upon which the hybrid system was based. These results and additional experiments demonstrate that RBF networks can be successfully incorporated in hybrid recognizers and suggest that they may be capable of good performance with fewer parameters than required by Gaussian mixture classifiers. A global parameter optimization method designed to minimize the overall word error rather than the frame recognition error failed to reduce the error rate.

## 1 HMM/RBF HYBRID RECOGNIZER

A hybrid isolated-word speech recognizer was developed which combines neural network and Hidden Markov Model (HMM) approaches. The hybrid approach is an attempt to capitalize on the superior static pattern classification performance of neural network classifiers [6] while preserving the temporal alignment properties of HMM Viterbi decoding. Our approach is unique when compared to other studies [2, 5] in that we use Radial Basis Function (RBF) rather than multilayer sigmoidal networks. RBF networks were chosen because their static pattern classification performance is comparable to that of other networks and they can be trained rapidly using a one-pass matrix inversion technique [8].

The hybrid HMM/RBF isolated-word recognizer is shown in Figure 1. For each

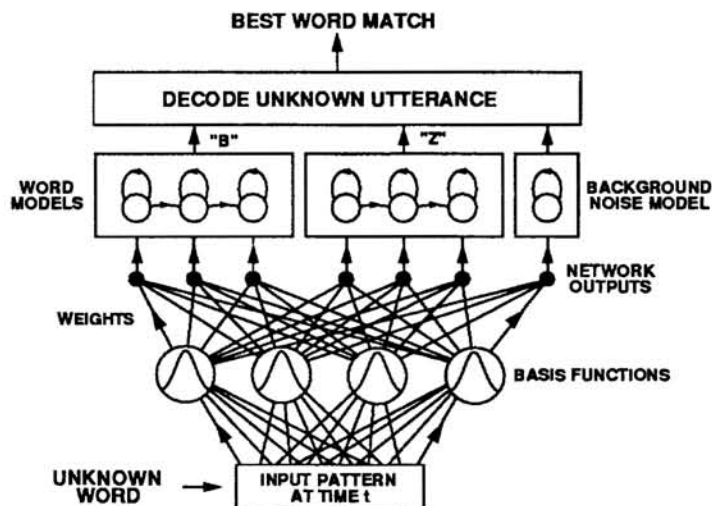

Figure 1: Block diagram of the hybrid recognizer for a two word vocabulary.

pattern presented at the input layer, the RBF network produces nodal outputs which are estimates of Bayesian probabilities [9]. The RBF network consists of an input layer, a hidden layer composed of Gaussian basis functions, and an output layer. Connections from the input layer to the hidden layer are fixed at unity while those from the hidden layer to the output layer are trained by minimizing the overall mean-square error between actual and desired output values. Each RBF output node has a corresponding state in a set of HMM word models which represent the words in the vocabulary. HMM word models are left-to-right with no skip states and have a one-state background noise model at either end. The background noise models are identical for all words. In the simplified diagram of Figure 1, the vocabulary consists of 2 E-set words and the HMMs contain 3 states per word model. The number of RBF output nodes (classes) is thus equal to the total number of HMM non-background states plus one to account for background noise. In recognition, Viterbi decoders use the nodal outputs of the RBF network as observation probabilities to produce word likelihood scores. Since the outputs of the RBF network can take on any value, they were initially hard limited to 0.0 and 1.0. The transition probabilities estimated as part of HMM training are retained. The final response of the recognizer corresponds to that word model which produces the highest Viterbi likelihood. Note that the structure of the HMM/RBF hybrid recognizer is identical to that of a tied-mixture HMM recognizer. For a discussion and comparison of the two recognizers, see [10].

Training of the hybrid recognizer begins with the preliminary step of training an HMM isolated-word recognizer. The robust HMM recognizer used provides good recognition performance on many standard difficult isolated-word speech databases [7]. It uses continuous density, unimodal diagonal-covariance Gaussian classifiers for each word state. Variances of all states are equal to the grand variance averaged over all words and states. The trained HMM recognizer is used to force an alignment of every training token and assign a label to each frame. Labels correspond to both states of HMM word models and output nodes of the RBF network.

The Gaussian centers in the RBF hidden layer are obtained by performing $k$-means

clustering on speech frames and separate clustering on noise frames, where speech and noise frames are distinguished on the basis of the initial Viterbi alignment. The RBF weights from the hidden layer to the output layer are computed by presenting input frames to the RBF network and setting the desired network outputs to 1.0 for the output node corresponding to the frame label and 0.0 for all other nodes. The RBF hidden node outputs and their correlations are accumulated across all training tokens and are used to estimate weights to the RBF output nodes using a fast one-pass algorithm [8]. Unlike the performance of the system reported in [5], additional training iterations using the hybrid recognizer to label frames did not improve performance.

## 2  DATABASE

All experiments were performed using a large, speaker-independent E-set (9 word) database derived from the ISOLET Spoken Letter Database [4]. The training set consisted of 1,080 tokens (120 tokens per word) spoken by 60 female and 60 male speakers for a total of 61,466 frames. The test set consisted of 540 tokens (60 tokens per word) spoken by a different set of 30 female and 30 male speakers for a total of 30,406 frames. Speech was sampled at 16 kHz and had an average SNR of 31.5 dB. Input vectors were based on a mel-cepstrum analysis of the speech waveform as described in [7]. The input analysis window was 20ms wide and was advanced at 10ms intervals. Input vectors were created by adjoining the first 12 non-energy cepstral coefficients, the first 13 first-difference cepstral coefficients, and the first 13 second-difference cepstral coefficients. Since the hybrid was based on an 8 state-per-word robust HMM recognizer, the RBF network contained a total of 73 output nodes (72 speech nodes and 1 background node). The error rate of the 8 state-per-word robust HMM recognizer on the speaker-independent E-set task was 15.7%.

## 3  MODIFICATIONS TO THE HYBRID RECOGNIZER

The performance of the baseline HMM/RBF hybrid recognizer described in Section 1 is quite poor. We found it necessary to select the recognizer structure carefully and utilize intermediate outputs properly to achieve a higher level of performance. A full description of these modifications is presented in [10]. Briefly, they include normalizing the hidden node outputs to sum to 1.0, normalizing the RBF outputs by the corresponding *a priori* class probabilities as estimated from the initial Viterbi alignment, expanding the RBF network into three individually trained subnetworks corresponding to the ceptrum, first difference cepstrum, and second difference cepstrum data streams, setting a lower limit of $10^{-5}$ on the values produced at the RBF output nodes, adjusting a global scaling factor applied to the variances of the RBF centers, and setting the number of centers to 33, 33, and 65 for the first, second, and third subnets, respectively. The structure of the final hybrid recognizer is shown in Figure 2. This recognizer has an error rate of 11.5% (binomial standard deviation = ±1.4) on the E-set test data compared to 15.7% (±1.6) for the 8 state-per-word unimodal Gaussian HMM recognizer, and 9.6% (±1.3) for a considerably more complex tied-mixture HMM recognizer [10]. The final hybrid system contained a total of 131 Gaussians and 9,563 weights. On a SUN SPARCstation 2, training time for

the final hybrid recognizer was about 1 hour and testing time was about 10 minutes.

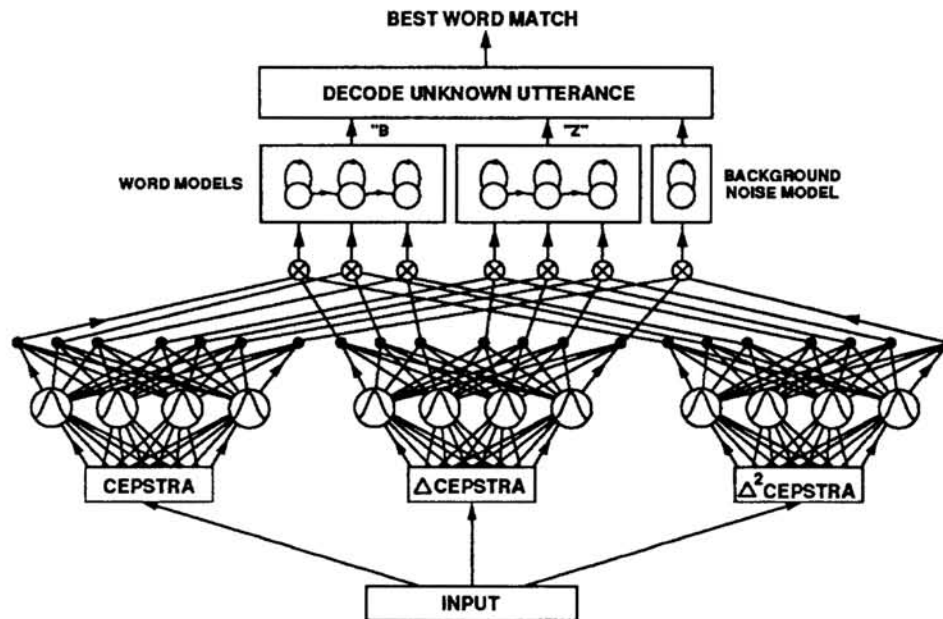

Figure 2: Block diagram of multiple subnet hybrid recognizer.

## 4    GLOBAL OPTIMIZATION

In the hybrid recognizer described above, discriminative training is performed at the frame level. A preliminary segmentation by the HMM recognizer assigns each speech frame to a specific RBF output node or, equivalently, an HMM word state. The RBF network weights are then computed to minimize the squared error between the network output and the desired output over all input frames. The goal of the recognizer, however, is to classify words. To meet this goal, discriminant training should be performed on word-level rather than frame-level outputs. Recently, several investigators have described techniques that optimize parameters based on word-level discriminant criteria [1, 3]. These techniques seek to maximize a mutual information type of criterion:

$$C = log \frac{L_c}{L},$$

where $L_c$ is the likelihood score of the word model corresponding to the correct result and $L = \sum_w L_w$ is the sum of the word likelihood scores for all models. By computing $\partial C/\partial \theta$, the gradient of $C$ with respect to parameter $\theta$, we can optimize any parameter in the hybrid recognizer using the update equation

$$\hat{\theta} = \theta + \eta \frac{\partial C}{\partial \theta},$$

where $\hat{\theta}$ is the new value of parameter $\theta$, $\theta$ is the previous value, and $\eta$ is a gain term proportional to the learning rate. Following [1], we refer to the word-level optimization technique as "global optimization."

To apply global optimization to the HMM/RBF hybrid recognizer, we derived the formulas for the gradient of $C$ with respect to $w_{ij}^k$, the weight connecting RBF center $i$ to RBF output node $j$ in subnet $k$; $p_j^k$, the RBF output normalization factor for RBF output node $j$ in subnet $k$; and $m_{il}^k$, the $l^{th}$ element of the mean of center $i$ of subnet $k$. For each token of length T frames, these are given by

$$\frac{\partial C}{\partial w_{ij}^k} = \left(\frac{\delta_{cj} - P_w}{L_w}\right) \sum_{t=1}^{T} \frac{\alpha_{jt}\beta_{jt}}{r_{jt}^k} \Phi_{it}^k,$$

$$\frac{\partial C}{\partial p_j^k} = \left(\frac{\delta_{cj} - P_w}{L_w}\right) \left(\frac{-1}{p_j^k}\right) \sum_{t=1}^{T} \alpha_{jt}\beta_{jt},$$

and

$$\frac{\partial C}{\partial m_{il}^k} = \sum_{j=1}^{N_k} \left(\frac{\delta_{cj} - P_w}{L_w}\right) w_{ij}^k \sum_{t=1}^{T} \frac{\alpha_{jt}\beta_{jt}}{r_{jt}^k} \left(\frac{x_{lt}^k - m_{il}^k}{h^k \sigma_{il}^k}\right) \Phi_{it}^k (1 - \Phi_{it}^k),$$

where

$$
\begin{aligned}
L_w &= \text{likelihood score for word model } w, \\
P_w &= L_w / \sum_w L_w \text{ is the normalized word likelihood,} \\
\delta_{cj} &= \begin{cases} 1 \text{ if RBF output node } j \text{ is a member of the correct word model} \\ 0 \text{ otherwise,} \end{cases} \\
\alpha_{jt} &= \text{forward partial probability of HMM state } j \text{ at time t,} \\
\beta_{jt} &= \text{backward partial probability of HMM state } j \text{ at time } t, \\
r_{jt}^k &= \text{unnormalized output of RBF node } j \text{ of subnet } k \text{ at time } t, \\
\Phi_{it}^k &= \text{normalized output of } i^{th} \text{ Gaussian center of subnet } k \text{ at time } t, \\
&\qquad \sum_i \Phi_{it}^k = 1, \\
x_{lt}^k &= l^{th} \text{ element of the input vector for subnet } k \text{ at time } t, \\
h^k &= \text{global scaling factor for the variances of subnet } k, \\
\sigma_{il}^k &= l^{th} \text{ component of the standard deviation of the } i^{th} \text{ Gaussian center} \\
&\qquad \text{of subnet } k, \\
N_k &= \text{number of RBF output nodes in subnet } k.
\end{aligned}
$$

In implementing global optimization, the frame-level training procedure described earlier serves to initialize system parameters and hill climbing methods are used to reestimate parameters iteratively. Thus, weights are initialized to the values derived using the one-pass matrix inversion procedure, RBF output normalization factors are initialized to the class priors, and Gaussian means are initialized to the $k$-means clustering values. Note that while the priors sum to one, no such constraint was placed on the RBF output normalization factors during global optimization.

It is worth noting that since the RBF network outputs in the hybrid recognizer are *a posteriori* probabilities normalized by *a priori* class probabilities, their values may exceed 1. The accumulation of these quantities in the Viterbi decoders often leads to values of $\alpha_{jt}\beta_{jt}$ and $L_w$ in the range of $10^{80}$ or greater. Numerical problems with the implementation of the global optimization equations were avoided by using log arithmetic for intermediate operations and working with the quantity $\beta_{jt}/L_w$ throughout. Values of $\eta$ which produced reasonable results were generally in the range of $10^{-10}$ to $10^{-6}$

The results of using the global optimization technique to estimate the RBF weights are shown in Figure 3. Figure 3(a) shows the recognition performance on the training and test sets versus the number of training iterations and Figure 3(b) tracks the value of the criterion $C = L_c/L$ on the training and test set under the same conditions. It is apparent that the method succeeds in iteratively increasing the value of the criterion and in significantly lowering the error rate on the training data. Unfortunately, this behavior does not extend to improved performance on the test data. This suggests that global optimization is overfitting the hybrid word models to the training data. Results using global optimization to estimate RBF output normalization factors and the Gaussian means produced similar results.

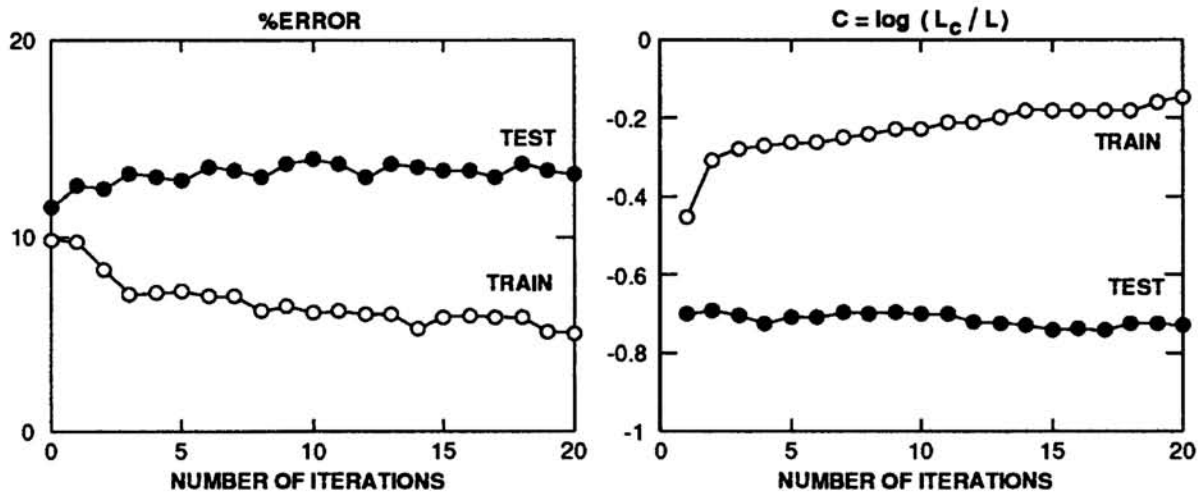

Figure 3: (a) Error rates for training and test data. (b) Criterion $C$ for training and test data.

# 5    ACCURACY OF BAYES PROBABILITY ESTIMATION

Three methods were used to determine how well RBF outputs estimate Bayes probabilities. First, since network outputs must sum to one if they are probabilities, we computed the RMS error between the sum of the RBF outputs and unity for all frames of the test data. The average RMS error was low ($10^{-4}$ or less for each subnet). Second, the average output of each RBF node was computed because this should equal the *a priori* probability of the class associated with the node [9]. This condition was true for each subnet with an average RMS error on the order of $10^{-5}$.

For the final method, we partitioned the outputs into 100 equal size bins between 0.0 and 1.0. For each input pattern, we used the output values to select the appropriate bins and incremented the corresponding bin counts by one. In addition, we incremented the correct-class bin count for the one bin which corresponded to the class of the input pattern. For example, data indicated that for the 61,466 frames of training tokens, nodal outputs of the cepstra subnet in the range 0.095-0.105 occurred 29,698 times and were correct classifications (regardless of class) 3,067 times. If the outputs of the network were true Bayesian probabilities, we would expect the

relative frequency of correct labeling to be close to 0.1. Similarly, relative frequencies measured in other intervals would also be expected to be close to the value of the corresponding center of the interval. Thus, a plot of the relative frequencies for each bin versus the bin centers should show the measured values lying close to the diagonal.

The measured relative frequency data for the cepstra subnet and $\pm 2\sigma$ bounds for the binomial standard deviations of the relative frequencies are shown in Figure 4. Outputs below 0.0 and above 1.0 are fixed at 0.0 and 1.0, respectively. Although the relative frequencies tend to be clustered around the diagonal, many values lie outside the bounds. Furthermore, goodness-of-fit measurements using the $\chi^2$ test indicate that fits fail at significance levels well below .01. We conclude that although the system provides good recognition accuracy, better performance may be obtained with improved estimation of Bayesian probabilities.

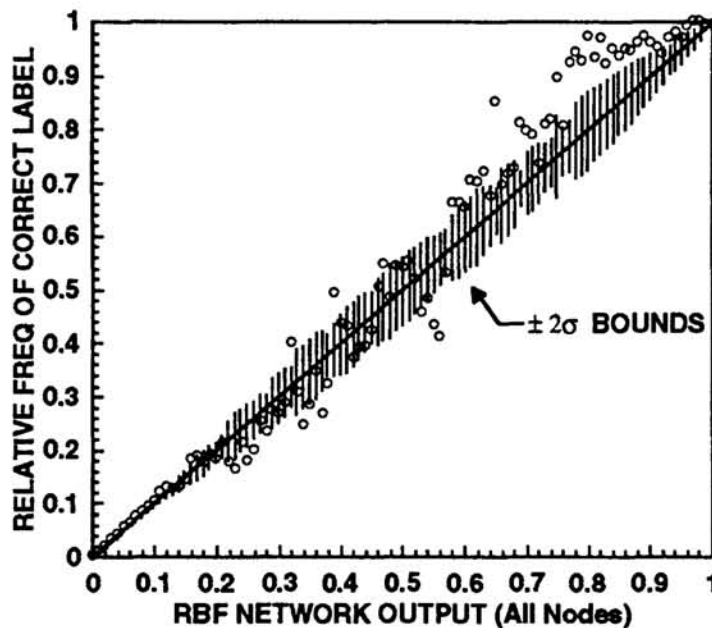

Figure 4: Relative frequency of correct class labeling and $\pm 2\sigma$ bounds for the binomial standard deviation, cepstra subnet.

## 6   SUMMARY AND CONCLUSIONS

This paper describes a hybrid isolated-word speech recognizer which successfully integrates Radial Basis Function neural networks and Hidden Markov Models. The hybrid's performance is better than that of a tied-mixture recognizer of comparable complexity and near that of a tied-mixture recognizer of considerably greater complexity. The structure of the RBF networks and the processing of network outputs had to be carefully selected to provide this level of performance. A global optimization technique designed to maximize a word discrimination criterion did not succeed in improving performance further. Statistical tests indicated that the accuracy of the Bayesian probability estimation performed by the RBF networks could

be improved. We conclude that RBF networks can be used to provide good performance and short training times in hybrid recognizers and that these systems may require fewer parameters than Gaussian-mixture-based recognizers at comparable performance levels.

## Acknowledgements

This work was sponsored by the Defense Advanced Research Projects Agency. The views expressed are those of the authors and do not reflect the official policy or position of the U.S. Government.

## References

[1] Yoshua Bengio, Renato De Mori, Giovanni Flammia, and Ralf Kompe. Global optimization of a neural network – Hidden Markov model hybrid. Technical Report TR-SOCS-90.22, MgGill University School of Computer Science, Montreal, Qc., Canada, December 1990.

[2] Herve Bourlard and Nelson Morgan. A continuous speech recognition system embedding MLP into HMM. In D. Touretzky, editor, *Advances in Neural Information Processing 2*, pages 186–193. Morgan Kaufmann, San Mateo, CA, 1990.

[3] John S. Bridle. Alpha-nets: A recurrent neural network architecture with a hidden Markov model interpretation. *Speech Communication*, 9:83–92, 1990.

[4] Ron Cole, Yeshwant Muthusamy, and Mark Fanty. The Isolet spoken letter database. Technical Report CSE 90-004, Oregon Graduate Institute of Science and Technology, Beverton, OR, March 1990.

[5] Michael Franzini, Kai-Fu Lee, and Alex Waibel. Connectionist viterbi training: A new hybrid method for continuous speech recognition. In *Proceedings of IEEE International Conference on Acoustics Speech and Signal Processing*. IEEE, April 1990.

[6] Richard P. Lippmann. Pattern classification using neural networks. *IEEE Communications Magazine*, 27(11):47–54, November 1989.

[7] Richard P. Lippmann and Ed A. Martin. Multi-style training for robust isolated-word speech recognition. In *Proceedings International Conference on Acoustics Speech and Signal Processing*, pages 705–708. IEEE, April 1987.

[8] Kenney Ng and Richard P. Lippmann. A comparative study of the practical characteristics of neural network and conventional pattern classifiers. In R. P. Lippmann, J. Moody, and D. S. Touretzky, editors, *Advances in Neural Information Processing 3*. Morgan Kaufmann, San Mateo, CA, 1991.

[9] Mike D. Richard and Richard P. Lippmann. Neural network classifiers estimate Bayesian a posteriori probabilities. *Neural Computation*, In Press.

[10] Elliot Singer and Richard P. Lippmann. A speech recognizer using radial basis function neural networks in an HMM framework. In *Proceedings of the International Conference on Acoustics, Speech, and Signal Processing*. IEEE, 1992.